# TOWARDS AN ORGANIZING PRINCIPLE FOR
# A LAYERED PERCEPTUAL NETWORK

Ralph Linsker
IBM Thomas J. Watson Research Center, Yorktown Heights, NY 10598

## Abstract

An information-theoretic optimization principle is proposed for the development of each processing stage of a multilayered perceptual network. This principle of "maximum information preservation" states that the signal transformation that is to be realized at each stage is one that maximizes the information that the output signal values (from that stage) convey about the input signals values (to that stage), subject to certain constraints and in the presence of processing noise. The quantity being maximized is a Shannon information rate. I provide motivation for this principle and -- for some simple model cases -- derive some of its consequences, discuss an algorithmic implementation, and show how the principle may lead to biologically relevant neural architectural features such as topographic maps, map distortions, orientation selectivity, and extraction of spatial and temporal signal correlations. A possible connection between this information-theoretic principle and a principle of minimum entropy production in nonequilibrium thermodynamics is suggested.

## Introduction

This paper describes some properties of a proposed information-theoretic organizing principle for the development of a layered perceptual network. The purpose of this paper is to provide an intuitive and qualitative understanding of how the principle leads to specific feature-analyzing properties and signal transformations in some simple model cases. More detailed analysis is required in order to apply the principle to cases involving more realistic patterns of signaling activity as well as specific constraints on network connectivity.

This section gives a brief summary of the results that motivated the formulation of the organizing principle, which I call the principle of "maximum information preservation." In later sections the principle is stated and its consequences studied.

In previous work[1] I analyzed the development of a layered network of model cells with feedforward connections whose strengths change in accordance with a Hebb-type synaptic modification rule. I found that this development process can produce cells that are selectively responsive to certain input features, and that these feature-analyzing properties become progressively more sophisticated as one proceeds to deeper cell layers. These properties include the analysis of contrast and of edge orientation, and are qualitatively similar to properties observed in the first several layers of the mammalian visual pathway.[2]

Why does this happen? Does a Hebb-type algorithm (which adjusts synaptic strengths depending upon correlations among signaling activities[3]) cause a developing perceptual network to optimize some property that is deeply connected with the mature network's functioning as an information processing system?

Further analysis[4,5] has shown that a suitable Hebb-type rule causes a linear-response cell in a layered feedforward network (without lateral connections) to develop so that the statistical variance of its output activity (in response to an ensemble of inputs from the previous layer) is maximized, subject to certain constraints. The mature cell thus performs an operation similar to principal component analysis (PCA), an approach used in statistics to expose regularities (e.g., clustering) present in high-dimensional input data. (Oja[6] had earlier demonstrated a particular form of Hebb-type rule that produces a model cell that implements PCA exactly.)

Furthermore, given a linear device that transforms inputs into an output, and given any particular output value, one can use optimal estimation theory to make a "best estimate" of the input values that gave rise to that output. Of all such devices, I have found that an appropriate Hebb-type rule generates that device for which this "best estimate" comes closest to matching the input values.[4,5] Under certain conditions, such a cell has the property that its output preserves the maximum amount of information about its input values.[5]

## Maximum Information Preservation

The above results have suggested a possible organizing principle for the development of each layer of a multilayered perceptual network.[5] The principle can be applied even if the cells of the network respond to their inputs in a nonlinear fashion, and even if lateral as well as feedforward connections are present. (Feedback from later to earlier layers, however, is absent from this formulation.) This principle of "maximum information preservation" states that for a layer of cells L that is connected to and provides input to another layer M, the connections should develop so that the transformation of signals from L to M (in the presence of processing noise) has the property that the set of output values M conveys the maximum amount of information about the input values L, subject to various constraints on, e.g., the range of lateral connections and the processing power of each cell. The statistical properties of the ensemble of inputs L are assumed stationary, and the particular L-to-M transformation that achieves this maximization depends on those statistical properties. The quantity being maximized is a Shannon information rate.[7]

An equivalent statement of this principle is: The L-to-M transformation is chosen so as to minimize the amount of information that would be conveyed by the input values L to someone who already knows the output values M.

We shall regard the set of input signal values L (at a given time) as an input "message"; the message is processed to give an output message M. Each message is in general a set of real-valued signal activities. Because noise is introduced during the processing, a given input message may generate any of a range of different output messages when processed by the same set of connections.

The Shannon information rate (i.e., the average information transmitted from L to M per message) is[7]

$$R = \Sigma_L \Sigma_M P(L,M) \log [P(L,M)/P(L)P(M)]. \tag{1}$$

For a discrete message space, $P(L)$ [resp. $P(M)$] is the probability of the input (resp. output) message being $L$ (resp. $M$), and $P(L,M)$ is the joint probability of the input being $L$ and the output being $M$. [For a continuous message space, probabilities are

replaced by probability densities, and sums (over states) by integrals.] This rate can be written as

$$R = I_L - I_{L|M} \tag{2}$$

where

$$I_L \equiv - \Sigma_L \, P(L) \log P(L) \tag{3}$$

is the average information conveyed by message $L$ and

$$I_{L|M} \equiv - \Sigma_M \, P(M) \, \Sigma_L \, P(L \,|\, M) \log P(L \,|\, M) \tag{4}$$

is the average information conveyed by message $L$ to someone who already knows $M$. Since $I_L$ is fixed by the properties of the input ensemble, maximizing $R$ means minimizing $I_{L|M}$, as stated above.

The information rate $R$ can also be written as

$$R = I_M - I_{M|L} \tag{5}$$

where $I_M$ and $I_{M|L}$ are defined by interchanging $L$ and $M$ in Eqns. 3 and 4. This form is heuristically useful, since it suggests that one can attempt to make $R$ large by (if possible) simultaneously making $I_M$ large and $I_{M|L}$ small. The term $I_M$ is largest when each message $M$ occurs with equal probability. The term $I_{M|L}$ is smallest when each $L$ is transformed into a unique $M$, and more generally is made small by "sharpening" the $P(M|L)$ distribution, so that for each $L$, $P(M|L)$ is near zero except for a small set of messages $M$.

How can one gain insight into biologically relevant properties of the $L \to M$ transformation that may follow from the principle of maximum information preservation (which we also call the "infomax" principle)? In a network, this $L \to M$ transformation may be a function of the values of one or a few variables (such as a connection strength) for each of the allowed connections between and within layers, and for each cell. The search space is quite large, particularly from the standpoint of gaining an intuitive or qualitative understanding of network behavior. We shall therefore consider a simple model in which the dimensionalities of the L and M signal spaces are greatly reduced, yet one for which the infomax analysis exhibits features that may also be important under more general conditions relevant to biological and synthetic network development.

The next four sections are organized as follows. (i) A model is introduced in which the L and M messages, and the L-to-M transformation, have simple forms. The infomax principle is found to be satisfied when some simple geometric conditions (on the transformation) are met. (ii) I relate this model to the analysis of signal processing and noise in an interconnection network. The formation of topographic maps is discussed. (iii) The model is applied to simplified versions of biologically relevant problems, such as the emergence of orientation selectivity. (iv) I show that the main properties of the infomax principle for this model can be realized by certain local algorithms that have been proposed to generate topographic maps using lateral interactions.

## A Simple Geometric Model

In this model, each input message $L$ is described by a point in a low-dimensional vector space, and the output message $M$ is one of a number of discrete states. For definiteness, we will take the L space to be two-dimensional (the extension to higher dimensionality is straightforward). The $L \rightarrow M$ transformation consists of two steps. (i) A noise process alters $L$ to a message $L'$ lying within a neighborhood of radius $\nu$ centered on $L$. (ii) The altered message $L'$ is mapped deterministically onto one of the output messages $M$.

A given $L' \rightarrow M$ mapping corresponds to a partitioning of the L space into regions labeled by the output states $M$. (We do not exclude a priori the possibility that multiple disjoint regions may be labeled by the same $M$.) Let $A$ denote the total area of the L state space. For each $M$, let $A(M)$ denote the area of L space that is labeled by $M$. Let $s(M)$ denote the total border length that the region(s) labeled $M$ share with regions of unlike M-label. A point $L$ lying within distance $\nu$ of a border can be mapped onto either M-value (because of the noise process $L \rightarrow L'$ ). Call this a "borderline" $L$. A point $L$ that is more than a distance $\nu$ from every border can only be mapped onto the M-value of the region containing it.

Suppose $\nu$ is sufficiently small that (for the partitionings of interest) the area occupied by borderline L states is small compared to the total area of the L space. Consider first the case in which $P(L)$ is uniform over L. Then the information rate $R$ (using Eqn. 5) is given approximately (through terms of order $\nu$) by

$$R = - \Sigma_M [A(M)/A] \log[A(M)/A] - (\gamma\nu/A) \Sigma_M s(M). \tag{6}$$

To see this, note that $P(M) = A(M)/A$ and that $P(M|L) \log P(M|L)$ is zero except for borderline $L$ (since $0 \log 0 = 1 \log 1 = 0$). Here $\gamma$ is a positive number whose value depends upon the details of the noise process, which determines $P(M|L)$ for borderline $L$ as a function of distance from the border.

For small $\nu$ (low noise) the first term $(I_M)$ on the RHS of Eqn. 6 dominates. It is maximized when the $A(M)$ [and hence the $P(M)$] values are equal for all $M$. The second term (with its minus sign), which equals $( -I_{M|L})$, is maximized when the sum of the border lengths of all $M$ regions is minimized. This corresponds to "sharpening" the $P(M|L)$ distribution in our earlier, more general, discussion. This suggests that the infomax solution is obtained by partitioning the L space into M-regions (one for each $M$ value) that are of substantially equal area, with each M-region tending to have near-minimum border length.

Although this simple analysis applies to the low-noise case, it is plausible that even when $\nu$ is comparable to the spatial scale of the M regions, infomax will favor making the M regions have approximately the same extent in all directions (rather than be elongated), in order to "sharpen" $P(M|L)$ and reduce the probability of the noise process mapping $L$ onto many different $M$ states.

What if $P(L)$ is nonuniform? Then the same result (equal areas, minimum border) is obtained except that both the area and border-length elements must now be weighted by the local value of $P(L)$. Therefore the infomax principle tends to produce maps in which greater representation in the output space is given to regions of the input signal space that are activated more frequently.

To see how lateral interactions within the M layer can affect these results, let us suppose that the $L \rightarrow M$ mapping has three, not two, process steps: $L \rightarrow L'$

$\rightarrow M' \rightarrow M$, where the first two steps are as above, and the third step changes the output $M'$ into any of a number of states $M$ (which by definition comprise the "M-neighborhood" of $M'$). We consider the case in which this M-neighborhood relation is symmetric.

This type of "lateral interaction" between $M$ states causes the infomax principle to favor solutions for which $M$ regions sharing a border in L space are M-neighbors in the sense defined. For a simple example in which each state $M'$ has $n$ M-neighbors (including itself), and each M-neighbor has an equal chance of being the final state (given $M'$), infomax tends to favor each M-neighborhood having similar extent in all directions (in L space).

## Relation Between the Geometric Model and Network Properties

The previous section dealt with certain classes of transformations from one message space to another, and made no specific reference to the implementation of these transformations by an interconnected network of processor cells. Here we show how some of the features discussed in the previous section are related to network properties.

For simplicity suppose that we have a two-dimensional layer of uniformly distributed cells, and that the signal activity of each cell at any given time is either 1 (active) or 0 (quiet). We need to specify the ensemble of input patterns. Let us first consider a simple case in which each pattern consists of a disk of activity of fixed radius, but arbitrary center position, against a quiet background. In this case the pattern is fully defined by specifying the coordinates of the disk center. In a two-dimensional L state space (previous section), each pattern would be represented by a point having those coordinates.

Now suppose that each input pattern consists not of a sharply defined disk of activity, but of a "fuzzy" disk whose boundary (and center position) are not sharply defined. [Such a pattern could be generated by choosing (from a specified distribution) a position $x_c$ as the nominal disk center, then setting the activity of the cell at position $x$ to 1 with a probability that decreases with distance $|x - x_c|$. ] Any such pattern can be described by giving the coordinates of the "center of activity" along with many other values describing (for example) various moments of the activity pattern relative to the center.

For the noise process $L \rightarrow L'$ we suppose that the activity of an L cell can be "misread" (by the cells of the M layer) with some probability. This set of distorted activity values is the "message" $L'$. We then suppose that the set of output activities $M$ is a deterministic function of $L'$.

We have constructed a situation in which (for an appropriate choice of noise level) two of the dimensions of the L state space -- namely, those defined by the disk center coordinates -- have large variance compared to the variance induced by the noise process, while the other dimensions have variance comparable to that induced by noise. In other words, the center position of a pattern is changed only a small amount by the noise process (compared to the typical difference between the center positions of two patterns), whereas the values of the other attributes of an input pattern differ as much from their noise-altered values as two typical input patterns differ from each other. (Those attributes are "lost in the noise.")

Since the distance between L states in our geometric model (previous section) corresponds to the likelihood of one L state being changed into the other by the noise

process, we can heuristically regard the L state space (for the present example) as a "slab" that is elongated in two dimensions and very thin in all other dimensions. (In general this space could have a much more complicated topology, and the noise process which we here treat as defining a simple metric structure on the L state space need not do so. These complications are beyond the scope of the present discussion.)

This example, while simple, illustrates a feature that is key to understanding the operation of the infomax principle: The character of the ensemble statistics and of the noise process jointly determine which attributes of the input pattern are statistically most significant; that is, have largest variance relative to the variance induced by noise. We shall see that the infomax principle selects a number of these most significant attributes to be encoded by the $L \rightarrow M$ transformation.

We turn now to a description of the output state space M. We shall assume that this space is also of low dimensionality. For example, each M pattern may also be a disk of activity having a center defined within some tolerance. A discrete set of discriminable center-coordinate values can then be used as the M-region "labels" in our geometric model.

Restricting the form of the output activity in this particular way restricts us to considering positional encodings $L \rightarrow M$, rather than encodings that make use of the shape of the output pattern, its detailed activity values, etc. However, this restriction on the form of the output does not determine which features of the input patterns are to be encoded, nor whether or not a topographic (neighbor-preserving) mapping is to be used. These properties will be seen to emerge from the operation of the infomax principle.

In the previous section we saw that the infomax principle will tend to lead to a partitioning of the L space into M regions having equal areas [if $P(L)$ is uniform in the coordinates of the L disk center] and minimum border length. For the present case this means that the M regions will tend to "tile" the two long dimensions of the L state space "slab," and that a single M value will represent all points in L space that differ only in their low-variance coordinates. If $P(L)$ is nonuniform, then the area of the M region at $L$ will tend to be inversely proportional to $P(L)$. Furthermore, if there are local lateral connections between M cells, then (depending upon the particular form of such interaction) M states corresponding to nearby localized regions of layer-M activity can be M-neighbors in the sense of the previous section. In this case the mapping from the two high-variance coordinates of L space to M space will tend to be topographic.

### Examples: Orientation Selectivity and Temporal Feature Maps

The simple example in the previous section illustrates how infomax can lead to topographic maps, and to map distortions [which provide greater M-space representation for regions of L having large $P(L)$]. Let us now consider a case in which information about input features is positionally encoded in the output layer as a result of the infomax principle.

Consider a model case in which an ensemble of patterns is presented to the input layer L. Each pattern consists of a rectangular bar of activity (of fixed length and width) against a quiet background. The bar's center position and orientation are chosen for each pattern from uniform distributions over some spatial interval for the position, and over all orientation angles (i.e., from $0°$ to $180°$). The bar need not be sharply defined, but can be "fuzzy" in the sense described above. We assume, however, that all

properties that distinguish different patterns of the ensemble -- except for center position and orientation -- are "lost in the noise" in the sense we discussed.

To simplify the representation of the solution, we further assume that only one coordinate is needed to describe the center position of the bar for the given ensemble. For example, the ensemble could consist of bar patterns all of which have the same $y$ coordinate of center position, but differ in their $x$ coordinate and in orientation $\theta$.

We can then represent each input state by a point in a rectangle (the L state space defined in a previous section) whose abscissa is the center-position coordinate $x$ and whose ordinate is the angle $\theta$. The horizontal sides of this rectangle are identified with each other, since orientations of $0°$ and $180°$ are identical. (The interior of the rectangle can thus be thought of as the surface of a horizontal cylinder.)

The number $N_x$ of different $x$ positions that are discriminable is given by the range of $x$ values in the input ensemble divided by the tolerance with which $x$ can be measured (given the noise process $L \rightarrow L'$); similarly for $N_\theta$. The relative lengths $\Delta x$ and $\Delta \theta$ of the sides of the L state space rectangle are given by $\Delta x / \Delta \theta = N_x / N_\theta$. We discuss below the case in which $N_x >> N_\theta$; if $N_\theta$ were $>> N_x$ the roles of $x$ and $\theta$ in the resulting mappings would be reversed.

There is one complicating feature that should be noted, although in the interest of clarity we will not include it in the present analysis. Two horizontal bar patterns that are displaced by a horizontal distance that is small compared with the bar length, are more likely to be rendered indiscriminable by the noise process than are two vertical bar patterns that are displaced by the same horizontal distance (which may be large compared with the bar's width). The Hamming distance, or number of binary activity values that need to be altered to change one such pattern into the other, is greater in the latter case than in the former. Therefore, the distance in L state space between the two

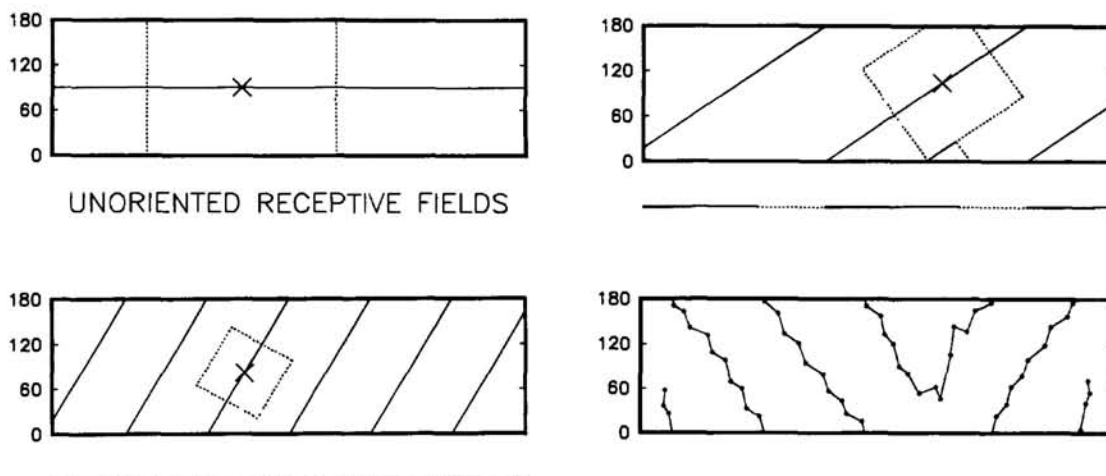

Figure 1.    Orientation Selectivity in a Simple Model:    As the input domain size (see text) is reduced [from (a) upper left, to (b) upper right, to (c) lower left figure], infomax favors the emergence of an orientation-selective $L \rightarrow M$ mapping. (d) Lower right figure shows a solution obtained by applying Kohonen's relaxation algorithm with 50 M-points (shown as dots) to this mapping problem.

states should be greater in the latter case. This leads to a "warped" rather than simple rectangular state space. We ignore this effect here, but it must be taken into account in a fuller treatment of the emergence of orientation selectivity.

Consider now an $L \rightarrow M$ transformation that consists of the three-step process (discussed above) (i) noise-induced $L \rightarrow L'$ ; (ii) deterministic $L' \rightarrow M'$; (iii) lateral-interaction-induced $M' \rightarrow M$. Step (ii) maps the two-dimensional L state space of points $(x, \theta)$ onto a one-dimensional M state space. For the present discussion, we consider $L' \rightarrow M'$ maps satisfying the following Ansatz: Points corresponding to the M states are spaced uniformly, and in topographic order, along a helical line in L state space (which we recall is represented by the surface of a horizontal cylinder). The pitch of the helix (or the slope $d\theta/dx$) remains to be determined by the infomax principle. Each M-neighborhood of M states (previous section) then corresponds to an interval on such a helix. A state $L'$ is mapped onto a state in a particular M-neighborhood if $L'$ is closer (in L space) to the corresponding interval of the helix than to any other portion of the helix. We call this set of L states (for an M-neighborhood centered on $M$ ) the "input domain" of $M$. It has rectangular shape and lies on the cylindrical surface of the L space.

We have seen (previous sections) that infomax tends to produce maps having (i) equal M-region areas, (ii) topographic organization, and (iii) an input domain (for each M-neighborhood) that has similar extent in all directions (in L space). Our choice of Ansatz enforces (i) and (ii) explicitly. Criterion (iii) is satisfied by choosing $d\theta/dx$ such that the input domain is square (for a given M-neighborhood size).

Figure 1a (having $d\theta/dx = 0$) shows a map in which the output $M$ encodes only information about bar center position $x$, and is independent of bar orientation $\theta$. The size of the M-neighborhood is relatively large in this case. The input domain of the state $M$ denoted by the 'x' is shown enclosed by dotted lines. (The particular $\theta$ value at which we chose to draw the M line in Fig. 1a is irrelevant.) For this M-neighborhood size, the length of the border of the input domain is as small as it can be.

As the M-neighborhood size is reduced, the dotted lines move closer together. A vertically oblong input domain (which would result if we kept $d\theta/dx = 0$ ) would not satisfy the infomax criterion. The helix for which the input domain is square (for this smaller choice of M-neighborhood size) is shown in Fig. 1b. The M states for this solution encode information about bar orientation as well as center position. If each M state corresponds to a localized output activity pattern centered at some position in a one-dimensional array of M cells, then this solution corresponds to orientation-selective cells organized in "orientation columns" (really "orientation intervals" in this one-dimensional model). A "labeling" of the linear array of cells according to whether their orientation preferences lie between 0 and 60, 60 and 120, or 120 and 180 degrees is indicated by the bold, light, and dotted line segments beneath the rectangle in Fig. 1b (and 1c).

As the M-neighborhood size is decreased still further, the mapping shown in Fig. 1c becomes favored over that of either Fig. 1a or 1b. The "orientation columns" shown in the lower portion of Fig. 1c are narrower than in Fig. 1b.

A more detailed analysis of the information rate function for various mappings confirms the main features we have here obtained by a simple geometric argument.

The same type of analysis can be applied to different types of input pattern ensembles. To give just one other example, consider a network that receives an ensemble of simple patterns of acoustic input. Each such pattern consists of a tone of

some frequency that is sensed by two "ears" with some interaural time delay. Suppose that the initial network layers organize the information from each ear (separately) into tonotopic maps, and that (by means of connections having a range of different time delays) the signals received by both ears over some time interval appear as patterns of cell activity at some intermediate layer L. We can then apply the infomax principle to the signal transformation from layer L to the next layer M. The L state space can (as before) be represented as a rectangle, whose axes are now frequency and interaural delay (rather than spatial position and bar orientation). Apart from certain differences (the density of L states may be nonuniform, and states at the top and bottom of the rectangle are no longer identical), the infomax analysis can be carried out as it was for the simplified case of orientation selectivity.

## Local Algorithms

The information rate (Eqn. 1), which the infomax principle states is to be maximized subject to constraints (and possibly as part of an optimization function containing other cost terms not discussed here), has a very complicated mathematical form. How might this optimization process, or an approximation to it, be implemented by a network of cells and connections each of which has limited computational power? The geometric form in which we have cast the infomax principle for some very simple model cases, suggests how this might be accomplished.

An algorithm due to Kohonen [8] demonstrates how topographic maps can emerge as a result of lateral interactions within the output layer. I applied this algorithm to a one-dimensional M layer and a two-dimensional L layer, using a Euclidean metric and imposing periodic boundary conditions on the short dimension of the L layer. A resulting map is shown in Fig. 1d. This map is very similar to those of Figs. 1b and 1c, except for one reversal of direction. The reversal is not surprising, since the algorithm involves only local moves (of the M-points) while the infomax principle calls for a globally optimal solution.

More generally, Kohonen's algorithm tends empirically [8] to produce maps having the property that if one constructs the Voronoi diagram corresponding to the positions of the M-points (that is, assigns each point $L$ to an M region based on which M-point $L$ is closest to), one obtains a set of M regions that tend to have areas inversely proportional to $P(L)$, and neighborhoods (corresponding to our input domains) that tend to have similar extent in all directions rather than being elongated.

The Kohonen algorithm makes no reference to noise, to information content, or even to an optimization principle. Nevertheless, it appears to implement, at least in a qualitative way, the geometric conditions that infomax imposes in some simple cases. This suggests that local algorithms along similar lines may be capable of implementing the infomax principle in more general situations.

Our geometric formulation of the infomax principle also suggests a connection with an algorithm proposed by von der Malsburg and Willshaw [9] to generate topographic maps. In their "tea trade" model, neighborhood relationships are postulated within the source and the target spaces, and the algorithm's operation leads to the establishment of a neighborhood-preserving mapping from source to target space. Such neighborhood relationships arise naturally in our analysis when the infomax principle is applied to our three-step $L \to L' \to M' \to M$ transformation. The noise process induces a

neighborhood relation on the L space, and lateral connections in the M cell layer can induce a neighborhood relation on the M space.

More recently, Durbin and Willshaw[10] have devised an approach to solving certain geometric optimization problems (such as the traveling salesman problem) by a gradient descent method bearing some similarity to Kohonen's algorithm.

There is a complementary relationship between the infomax principle and a local algorithm that may be found to implement it. On the one hand, the principle may explain what the algorithm is "for" -- that is, how the algorithm may contribute to the generation of a useful perceptual system. This in turn can shed light on the system-level role of lateral connections and synaptic modification mechanisms in biological networks. On the other hand, the existence of such a local algorithm is important for demonstrating that a network of relatively simple processors -- biological or synthetic -- can in fact find global near-maxima of the Shannon information rate.

## A Possible Connection Between Infomax and a Thermodynamic Principle

The principle of "maximum preservation of information" can be viewed equivalently as a principle of "minimum dissipation of information." When the principle is satisfied, the loss of information from layer to layer is minimized, and the flow of information is in this sense as "nearly reversible" as the constraints allow. There is a resemblance between this principle and the principle of "minimum entropy production" [11] in nonequilibrium thermodynamics. It has been suggested by Prigogine and others that the latter principle is important for understanding self-organization in complex systems. There is also a resemblance, at the algorithmic level, between a Hebb-type modification rule and the autocatalytic processes[12] considered in certain models of evolution and natural selection. This raises the possibility that the connection I have drawn between synaptic modification rules and an information-theoretic optimization principle may be an example of a more general relationship that is important for the emergence of complex and apparently "goal-oriented" structures and behaviors from relatively simple local interactions, in both neural and non-neural systems.

## References

[1]    R. Linsker, *Proc. Natl. Acad. Sci. USA* **83** , 7508, 8390, 8779 (1986).
[2]    D. H. Hubel and T. N. Wiesel, *Proc. Roy. Soc. London* **B198** , 1 (1977).
[3]    D. O. Hebb, *The Organization of Behavior* (Wiley, N. Y., 1949).
[4]    R. Linsker, in: R. Cotterill (ed.), *Computer Simulation in Brain Science* (Copenhagen, 20-22 August 1986; Cambridge Univ. Press, in press), p. 416.
[5]    R. Linsker, *Computer* (March 1988, in press).
[6]    E. Oja, *J. Math. Biol.* **15** , 267 (1982).
[7]    C. E. Shannon, *Bell Syst. Tech. J.* **27** , 623 (1948).
[8]    T. Kohonen, *Self-Organization and Associative Memory* (Springer-Verlag, N. Y., 1984).
[9]    C. von der Malsburg and D. J. Willshaw, *Proc. Natl. Acad. Sci. USA* **74** , 5176 (1977).
[10]    R. Durbin and D. J. Willshaw, *Nature* **326** , 689 (1987).
[11]    P. Glansdorff and I. Prigogine, *Thermodynamic Theory of Structure, Stability, and Fluctuations* (Wiley-Interscience, N. Y., 1971).
[12]    M. Eigen and P. Schuster, *Die Naturwissenschaften* **64** , 541 (1977).
